# ReMAP: Neural Model Reprogramming with Network Inversion and Retrieval-Augmented Mapping for Adaptive Motion Forecasting

**Sharmita Dey**\*, **Sarath Ravindran Nair**\*

## Abstract

Mobility impairment caused by limb loss, aging, stroke, and other movement deficiencies is a significant challenge faced by millions of individuals worldwide. Advanced assistive technologies, such as prostheses and orthoses, have the potential to greatly improve the quality of life for such individuals. A critical component in the design of these technologies is the accurate forecasting of reference joint motion for impaired limbs, which is hindered by the scarcity of joint locomotion data available for these patients. To address this, we propose ReMAP, a novel model repurposing strategy that leverages deep learning's reprogramming property, incorporating network inversion principles and retrieval-augmented mapping. Our approach adapts models originally designed for able-bodied individuals to forecast joint motion in limb-impaired patients without altering model parameters. We demonstrate the efficacy of ReMAP through extensive empirical studies on data from below-knee-challenged patients, showcasing significant improvements over traditional transfer learning and fine-tuning methods. These findings have significant implications for advancing assistive technology and mobility for patients with amputations, stroke, or aging.

## 1   Introduction

Physical impairment is a life-altering event that affects millions of individuals worldwide, imposing substantial challenges on their mobility and daily activities. The disability of a limb can lead to significant physical and psychological limitations, impacting the individual's overall well-being and independence. In recent years, considerable efforts have been made to develop advanced assistive technologies to address these challenges and enhance the quality of life for impaired patients.

A crucial aspect of designing effective assistive technologies is the accurate prediction of reference joint motion for the impaired limb. Understanding the natural motion of the joints is essential for the development of prosthetic devices that can restore the function of the impaired limb seamlessly. However, obtaining reliable data for impaired patients is a complex task, as their numbers and ability to perform diverse motion conditions are relatively limited compared to able-bodied individuals. This scarcity of data hinders the training of robust models specifically tailored to the unique conditions of each individual. Moreover, each impairment is a unique and individualized event, leading to a wide range of motion patterns and functional variations among such patients. Consequently, creating a single generic model for those patients is not practical, as it would not capture the individual variations

Sharmita Dey is with the University Medical Center Göttingen, Georg-August University of Göttingen sharmita.dey@med.uni-goettingen.de

Sarath Ravindran Nair is with the Georg-August University of Göttingen, Germany s.ravindrannair@eni-g.de

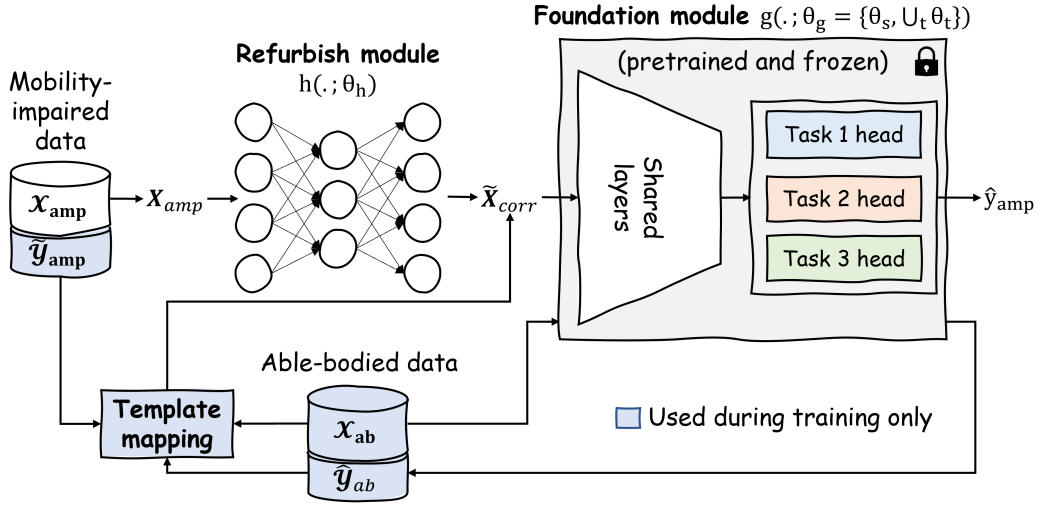

Figure 1: Simplified architecture of the proposed ReMAP. The corrupt inputs $X_{amp}$ from the individuals with mobility challenges are mapped to clean inputs $X_{corr}$ computed from able-bodied individuals, and the corrected inputs are used to produce the desired motion variables for the individuals with mobility challenges $\hat{y}_{amp}$ using a frozen foundation module pretrained for able-bodied subjects.

and diverse motion conditions that arise from different types of impairments. In contrast, subject-specific modeling attempts to cater to each patient's specific needs and characteristics. However, this approach faces significant challenges as well. The limited data availability for individual patients poses a hurdle in developing accurate models, leading to suboptimal performance and generalization issues.

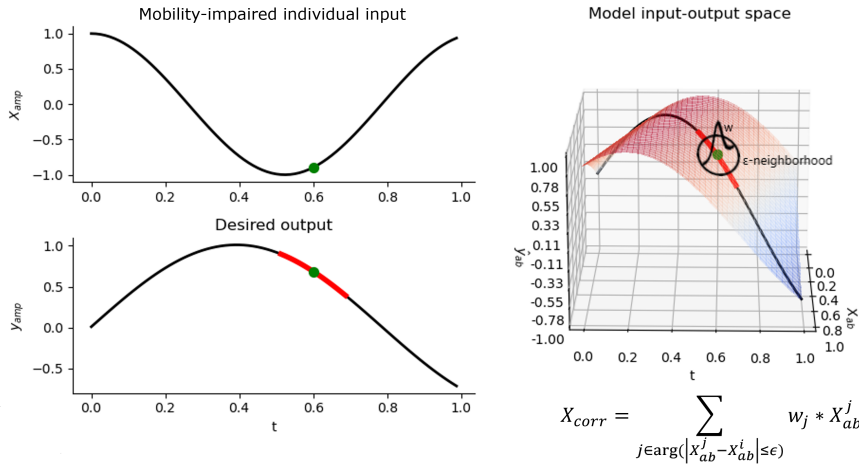

$$X_{corr} = \sum_{j \in \arg(|X_{ab}^j - X_{ab}^i| \le \epsilon)} w_j * X_{ab}^j$$

Figure 2: Illustration of computation of the correction input $X_{corr}$ corresponding to the $k$-th input sample $X_{amp}^k$ of the mobility-impaired individual. The able-bodied input $X_{ab}^i$ that produces the most similar output as that of the desired mobility-impaired individual output $y_{amp}^k$ is searched in the input-output space of the trained able-bodied foundation module. Instead of searching based on a single desired motion variable $y_{amp}^k$, a sequence of values $\{y_{amp}^{k-m}, ..., y_{amp}^k, ..., y_{amp}^{k+m}\}$ (marked by the red region in the lower left curve) is used and the able-bodied input $X_{ab}^i$ corresponding to the midpoint of the sequence is considered. Further, a neighborhood of radius $\epsilon$ is considered around $X_{ab}^i$ and the correction input $X_{corr}$ is computed as a weighted sum of samples in this neighborhood with weights decreasing (linearly or exponentially) with increasing distance from the center $X_{ab}^i$.

In this research, we introduce ReMAP, a novel approach for motion generation in impaired individuals using neural model reprogramming. Our strategy addresses the challenges of predicting joint motion with limited data by repurposing well-trained gait prediction models developed on able-bodied data. ReMAP utilizes network inversion principles and retrieval-augmented mapping to adapt models for impaired individuals without requiring retraining or fine-tuning of the pretrained models. ReMAP comprises three key components: (1) a diverse foundation module derived from able-bodied data, (2) a retrieval-augmented template mapping module to identify the most relevant inputs for learning the mapping, and (3) a refurbish module that learns the mapping once the desired inputs are identified. Specifically, we leverage network inversion techniques [1, 2, 3] to generate correction templates for mapping impaired inputs to able-bodied patterns and retrieval-based methods [4, 5, 6, 7] to identify the most relevant inputs for learning this mapping.

Furthermore, the adoption of neural model reprogramming provides an additional advantage in terms of cost and resource efficiency. Fine-tuning subject-specific models can be computationally expensive, requiring modifications to a considerable number of parameters in the model. In contrast, model reprogramming optimizes a smaller set of parameters, reducing the computational overhead significantly. Additionally, this technique preserves the model's original functionality, making it adaptable to various tasks without compromising its initial training.

We evaluate our method against baselines using transfer learning and conduct extensive ablations on various reprogramming techniques and architectures. Our quantitative results demonstrate the effectiveness of our approach, particularly in very low-data regimes, where it outperforms all tested transfer learning and fine-tuning methods. To the best of our knowledge, this is the first work in the domain of motion regression using neural model reprogramming. Our findings have significant implications for the advancement of assistive technologies, offering the potential to improve mobility and quality of life for limb-impaired patients.

## 2 Related Work

**Reprogramming of models.** The concept of model reprogramming involves repurposing proficiently trained models for novel tasks through data-level manipulation alone [8]. This underscores the capability of deep models to handle diverse tasks without the need to alter any of their model parameters. The efficiency of this reprogramming approach has been validated in the context of image classification [8, 9], time-series classification [10, 11], spoken command recognition [12], GAN conditioning [13], out-of-distribution detection [14], antibody sequence filling [15] and fraud detection [16]. This research explores the potential of model reprogramming to address the challenges of predicting joint motion in lower-limb-impaired individuals, aiming to enhance the development of assistive technologies. By repurposing well-established gait prediction models trained on able-bodied data, we seek to provide limb-impaired individuals with improved mobility solutions. This approach has the potential to transform the landscape of prosthetic development, offering a higher quality of life for individuals facing mobility challenges.

**Network inversion.** Network inversion techniques have been instrumental in optimizing neural network inputs to achieve specified outputs. Initially proposed by Linden and Kindermann [1], this method utilizes gradient-based optimization to iteratively refine inputs until the network outputs the desired result [1, 2, 3]. This approach has been studied and applied in interpretability research, such as visualizing deep convolutional networks to find the input image that maximizes the activation of a particular neuron or layer [17], understanding deep image representations [18, 19, 20], and image synthesis [21]. In our work, we leverage this concept to generate a correction template for mapping inputs, given the characteristics of the desired output motion.

**Retrieval-based methods.** Retrieval-based methods rely on retrieving relevant information from a large corpus in response to a query [4, 5, 6, 7]. These methods involve identifying and retrieving relevant documents or passages using techniques like dense retrieval [22] and using this information to inform subsequent processes such as text generation or model training. A notable application of retrieval-based methods is Retrieval-Augmented Generation (RAG), where retrieval provides context that enhances generative processes [23, 24]. We apply retrieval-based methods to identify segments of impaired-limb motion dynamics, to learn a neural reprogramming module.

**Gait motion models.** Gait prediction is a challenging problem due to the complex nature of human gait. In recent years, there has been a growing interest in developing gait prediction models

using machine learning techniques. Most works focused on developing such models by training explicit models to learn the input-output synergy, especially, for able-bodied subjects [25, 26, 27, 28, 29, 30, 31], with only a few works on lower-limb-impaired individuals [32, 33, 34]. Directly applying a gait prediction model trained for able-bodied subjects for motion prediction in impaired individuals suffered in performance due to the inherent differences between gait patterns of able-bodied individuals and limb-impaired individuals [34]. This study presents an efficient strategy to adapt a generic gait prediction model for predicting limb joint motion during various locomotion tasks for below-knee impaired individuals, including walking and stair ascent/descent, without requiring model fine-tuning or retraining. Our method integrates retrieval with network inversion techniques [2, 3], mapping impaired inputs to able-bodied patterns to facilitate accurate motion prediction.

## 3 Datasets

**Able-bodied dataset.** To train the able-bodied motion model, we utilize a comprehensive public dataset [35] that includes kinematic data collected from wearable sensors. These sensors comprise IMUs that measure 3D angular velocities and linear accelerations, as well as goniometers that capture sagittal knee and ankle positions under various motion conditions. The dataset includes recordings from ten individuals performing a range of locomotion activities, such as level ground walking, stair ascent, stair descent, and walking on inclined walkways. The dataset comprises approximately 6.56 million samples across ten individuals, averaging about 656,039 samples per individual.

**Lower-limb impairment motion datasets.** Motion data were collected from individuals with below-knee lower-limb impairments/amputations [36] using a 200 Hz camera-based motion capture system (Vicon Motion Systems Ltd., UK) equipped with 12 cameras. Retro-reflective markers were placed on their bony landmarks in the torso, pelvis, thigh, shank, and residual foot, with additional markers on the thigh and shank for 3D tracking. Various locomotion tasks were performed, including different walking speeds, stair ascent, and stair descent. Gait event detection software (Vicon Nexus) marked gait cycle boundaries based on marker positions and force thresholds. We processed the marker trajectories and computed joint angles and kinematic data using OpenSim [37], an open-source musculoskeletal modeling platform. The user studies were approved by the Institutional Review Board (IRB) of the University Medical Center Göttingen, Germany. Participants were informed beforehand about the experimental procedure, potential risks, precaution measures, and data protection. Experiments were conducted after obtaining written consent from participants. All ethical protocols regarding information, instructions, and compensation were followed. Details are provided in the appendix A.5.

**Inputs and outputs.** We utilized the temporal history of angular velocities from the shank and thigh segments and the angular position of the knee joint on both sides as model inputs.

$$X_t = \{\theta_{t-K:t}^{(thigh,r)}, \theta_{t-K:t}^{(thigh,l)}, \theta_{t-K:t}^{(shank,r)}, \theta_{t-K:t}^{(shank,l)}, \theta_{t-K:t}^{(knee,r)}, \theta_{t-K:t}^{(knee,l)}\} \in \mathcal{R}^{K \times D} \tag{1}$$

where $K = 20$ is the length of history and $D = 6$ is the number of input features. The model predicted the angular position of the ankle joint on the impaired side. Given that direct measurement of ground truth outputs from lower-limb impaired individuals is not feasible due to limb disabilities or loss, we computed the desired ankle motion trajectories for each such individual's gait cycle based on similar-speed gait cycles of a subset of able-bodied subjects with comparable anthropometric features (mass, age, height).

$$\tilde{y}_{amp} = \frac{\sum_{s \in S_{anthropometric}} \sum_{n \in N_{speed}} y_k^{s,n}}{|S_{anthropometric}||N_{speed}|} \tag{2}$$

where $S_{anthropometric}$ is the set of subjects with similar anthropometry as the impaired individual (height: ±5cm, weight: ±5kg) and $N_{apeed}$ is the set of gait cycles where the able-bodied subjects in walked at similar speeds (±0.1m/s) as that of the impaired individual. The reference timepoint $k$ for computing a matching output for the impaired individual was based on the phase of the gait cycle.

# 4 Method

Our objective is to utilize a well-trained foundation model that has learned a variety of motion scenarios from physically capable individuals to predict the walking patterns of a lower-limb impaired individual's disabled limbs, all without fine-tuning or altering the model parameters. This involves reprogramming/correcting the inputs acquired from the impaired individuals by refurbishing them so that the foundation model is able to produce the desired motion variables from the limb-impaired individuals' inputs. Our method integrates retrieval with network inversion techniques, efficiently mapping impaired-individual inputs to able-bodied patterns and facilitating accurate motion prediction. It comprises three components: a *foundation* module, a versatile multi-task model developed from data sourced from numerous able-bodied subjects; a *retrieval-augmented template mapping* module, which computes the correction template for the queried impaired individual input; and a *refurbish* module, designed to map the queried impaired individual input to the computed correction template. The *foundation* module then predicts the desired motion of the disabled limb from the refurbished impaired individuals' inputs. Figure 1 summarizes this architecture. Below, we elaborate on each of the components.

## 4.1 Foundation module

We train a foundation multi-task module using data from able-bodied individuals $(\mathcal{X}_{ab}, \mathcal{Y}_{ab})$, encompassing the different motion conditions. A fundamental difficulty in the realm of multi-task learning is finding a balance between utilizing the common patterns among multiple tasks while still preserving the adaptability needed to address the unique characteristics of each individual task. Our strategy involves constructing a predictive model, denoted as $g(.)$, which comprises two distinct components: a shared core $g_s$ with parameters $\theta_s$ that are shared across all tasks, and subsequently, task-specific layers $g_t$ with parameters $\theta_t$. The holistic forecast for a given input instance $X_{t,k}$, pertaining to the task $t$ at the time point $k$, is represented as $g(X_{t,k}, t; \theta_g = \{\theta_s, \bigcup_t \theta_t\}) = g_t(g_s(X_{t,k}; \theta_s); \theta_t)$. For the shared core $g_s$, we employ time convolutions [38]. We deploy lightweight task-specific layers, $g_t$, on top of the shared core, characterized by a two-layer multi-layer perceptron (MLP) with rectified linear unit (ReLU) nonlinearities. We compare architectures with task-specific final layers to a model without such task-specific parameters. In this alternate architecture, the output features $g_s(X; \theta_s)$ of the shared core for any task are passed through a single prediction head $g_c$ to produce the output motion variable. This comparison helps evaluate the efficacy of task-specific adaptations versus a unified approach in predicting motion variables across different tasks.

## 4.2 Retrieval-augmented template mapping

Once the able-bodied model is trained, we use it to predict the motion variables for the lower-limb mobility-impaired subjects without retraining the model with the affected individual's data. To achieve this, we map the mobility-impaired individual's inputs $X_{amp}$ corresponding to a desired output $y_{amp}$ to a corrected input $X_{corr}$ such that the model gives a similar output as the mobility-impaired individual's desired output for this corrected input (Fig. 1). We identify two methods for computing such a correction input $X_{corr}$, namely *nearest neighbor search* and *network inversion*.

**Nearest neighbor search.** A naive way of mapping the impaired individual's input to an able-bodied template is to use the able-bodied input $X_{ab}$ for which the model gives the desired impaired individual's output $\tilde{y}_{amp}^k$ as the corrected input, that is,

$$X_{corr}^k = \underset{X_{ab} \in \mathcal{X}_{ab}}{\arg\min} \|f(X_{ab}) - \tilde{y}_{amp}^k\| \tag{3}$$

, where the desired impaired individual's output $\tilde{y}_{amp}^k$ is computed from those able-bodied individuals sharing similar anthropometry (such as height, mass, and age) as that of the impaired individual.

However, an inherent problem of this approach is the ambiguity in the corrected input when similar values of desired output $y_{amp}$ occur separated in time. This can lead to entirely different input values being mapped as correction inputs for similar input values of the limb-impaired individuals. For example, in the illustration in Fig. 2, consider we were searching for a $X_{corr}$ for $y_{amp}^k = 0.6$ at $t = 0.6$. However, $y_{amp}^k = 0.6$ also occurs around $t = 0.2$ for which the correction input might be completely different from the one at $t = 0.6$.

One way to disentangle this problem would be to consider not a single $y_{amp}^k$ value for computing the correction input, but a sequence of co-occurring values in a time of which the desired output $y_{amp}^k$ is the midpoint (red region in Fig. 2). The algorithm now searches for a sequence of $2m+1$ able-bodied inputs that would produce the desired sequence of mobility-impaired individual outputs $\{y_{amp}^{k-m}, ..., y_{amp}^k, ..., y_{amp}^{k+m}\}$. This way, it becomes less probable that different able-bodied input sequences correspond to the same mobility-impaired individual output sequences. For example, the desired output $y_{amp}^k$ at $t = 0.6$ in Fig. 2 falls in the region of decreasing values of $y_{amp}$, whereas the other similar value at around $t = 0.2$ lies in the region of increasing $y_{amp}$ values. These two output values, although similar, would thus have different able-bodied inputs associated with them for correction. The correction input $X_{corr}^k$ in this case can be computed as

$$X_{corr}^k = \arg\min_{X_{ab}^i} \sum_{j=-m}^{m} \|f(X_{ab}^{i-j}) - \tilde{y}_{amp}^{k-j}\| \tag{4}$$

In the previous approaches, we used a single value of able-bodied input as a correction template for the mobility-impaired individual. However, this approach may be prone to noise and overfitting. To deal with this, we propose an extension of our approach which integrates multiple able-bodied input values to form a correction template for the mobility-impaired individual. This is achieved by defining a $\epsilon$-neighborhood (black ellipse in Fig. 1) around the able-bodied input for which the model prediction is closest to the desired mobility-impaired individual output (such an able-bodied input is obtained using either Eqn. 3 or Eqn. 4). The correction template is computed as a weighted sum of the able-bodied inputs within the $\epsilon$-neighborhood. If $X_{ab}^i$ be the closest able-bodied input around which the $\epsilon$-neighborhood is defined, the correction template is given by

$$X_{corr}^k = \sum_{j \in arg(\|X_{ab} - X_{ab}^i\| \leq \epsilon)} w_j * X_{ab}^j. \tag{5}$$

where $w_j$ is the weight associated with each of the able-bodied input $X_{ab}^j$ in the $\epsilon$-neighborhood. We define two types of weighting – linear and exponential – with the weight $w_j$ of an able-bodied input sample $X_{ab}^j$ decreasing linearly or exponentially as it moves away from the center $X_{ab}^i$ of the $\epsilon$-neighborhood. The weighting factor $w_j$ is thus given by

$$w_j = \begin{cases} 1 - \frac{X_{ab}^j - X_{ab}^i}{\epsilon} & \text{if weighting = linear} \\ exp(s * \frac{X_{ab}^i - X_{ab}^j}{\epsilon}) & \text{if weighting = exponential} \end{cases} \tag{6}$$

where $s$ is a scaling factor that determines how fast the weight decreases as the sample $X_{ab}^j$ moves away from the center of the $\epsilon$-neighborhood. Other types of weighting (for example, uniform weighting of samples in the neighborhood) are also possible, but are out of scope of this work.

For simplicity, we consider only $n$ closest neighbors within the $\epsilon$-neighborhood for computing the correction template $X_{corr}$. Since the inputs are normalized in $[0, 1]$, $\epsilon$ is set to 0.01.

**Network inversion.** One limitation of the nearest neighbor search method detailed above is the requirement to store the complete training data used for training the foundation module, which increases the memory requirements. Although one can devise clever ways to store less data without affecting the accuracy, we propose directly retrieving the correction input $\mathbf{X}_{corr}$ corresponding to the mobility-impaired individual's desired output, $\tilde{y}_{amp}$ from the pretrained foundation module. For the desired mobility-impaired individual output, $y_{amp}^k$ at each time point $k$, we generate a correction template $X_{corr}^k$ such that,

$$X_{corr}^k = \arg\min_X \left\|g(X) - \tilde{y}_{amp}^k\right\|_2 + \lambda \left\|X_{corr}^k\right\|_2 \tag{7}$$

where the $\lambda$ is the regularization strength.

This approach draws inspiration from network-inversion techniques introduced by Linden and Kindermann [1] which involves finding a relevant input that produces a specific output from a neural network. This technique, leverages gradient-based optimization to iteratively adjust the input until the desired output is achieved.

## 4.3 Refurbish module

We perform a data-level adaptation of the mobility-impaired individual inputs by learning a mapping from the corrupted mobility-impaired individual data $X_{amp}$ (due to compensatory motion, asymmetric gait) to the correction template $X_{corr}$ computed from the able-bodied data using the techniques mentioned in the previous section. To achieve this, we train a lightweight refurbish module $h(.)$ characterized by a multi-layer perceptron with three hidden layers of 100 units each. Following three strategies are employed for training the refurbish module.

**Correction-based mapping.** The refurbish module $h(.)$ is trained to map the mobility-impaired individual input $X_{amp}$ to the corresponding computed correction $X_{corr}$ by minimizing the MSE loss between the computed correction $X_{corr}$ and refurbish module predicted correction $\hat{X}_{corr} = h(X_{amp}; \Theta_h)$. The loss function is

$$\mathcal{L}_{input} = \sum_{X_{amp}, X_{corr} \in \mathcal{X}_{amp}^{train}} \|h(X_{amp}; \Theta_h) - X_{corr}\| \tag{8}$$

where $\mathcal{X}_{amp}^{train}$ is the training dataset consisting of mobility-impaired individual inputs and corresponding correction templates. The correction template $X_{corr}$ may be computed using either nearest *neighbor search* or *network inversion*, detailed in the previous section.

**Target-based mapping.** In this case, the input to map to is not explicitly determined, instead the refurbish module optimizes its parameters to produce the target directly, given the frozen foundation module. The refurbish module $h(.)$ is trained to minimize the MSE loss between the predictions of the pretrained foundation module $g(.)$ when it is fed with the refurbish module's outputs and the desired target values of the mobility-impaired individual $y_{amp}$. In this strategy, the computed correction templates are completely ignored, and the model is trained solely to reproduce the final desired target. The loss function becomes

$$\mathcal{L}_{target} = \sum_{X_{amp}, y_{amp} \in \mathcal{X}_{amp}^{train}, \tilde{\mathcal{Y}}_{amp}^{train}} \|g(h(X_{amp}; \Theta_h); \Theta_g^*) - \tilde{y}_{amp}\| \tag{9}$$

where $\tilde{\mathcal{Y}}_{amp}^{train}$ is the training dataset consisting of desired outputs for mobility-impaired individuals. The foundation module $g(.)$ is pretrained using the data from multiple able-bodied subjects, and its parameters $\Theta_g^*$ are kept fixed. Only the parameters $\Theta_h$ of the refurbish module $h(.)$ are varied using this loss function.

**Hybrid.** This is a combination of the correction-based and target-based training strategies. The loss function used to train the mapping model $h$ is a weighted combination of the error between the predicted correction $\hat{X}_{corr} = h(X_{amp})$ for the mobility-impaired individual input $X_{amp}$ and its computed correction $X_{corr}$ and the error between foundation module predictions on mapping model outputs, $\hat{y}_{amp} = g(h(X_{amp}))$ and the desired output values of the mobility-impaired individual $\tilde{y}_{amp}$. Thus, the hybrid loss function is,

$$\mathcal{L}_{total} = \alpha * \mathcal{L}_{input} + \beta * \mathcal{L}_{target} \tag{10}$$

where $\alpha$ and $\beta$ are the factors that influence the effect of input-based loss and target-based loss respectively on the final loss. Based on which correction strategy is used, the hybrid strategy can take two forms – **hybrid (neighbor)** and **hybrid (inversion)**.

## 5 Experiments

### 5.1 Baselines

We assess our proposed method's effectiveness against three baseline approaches:

**Cross-mapping (zero-shot transfer).** Here, we directly apply the foundation model $g(.)$ to predict motion variables for mobility-impaired individual subjects, without retraining the foundation module or utilizing the refurbish module to transform mobility-impaired individual inputs into clean data. This is akin to zero-shot transfer, where the pretrained model is used to predict output trajectories

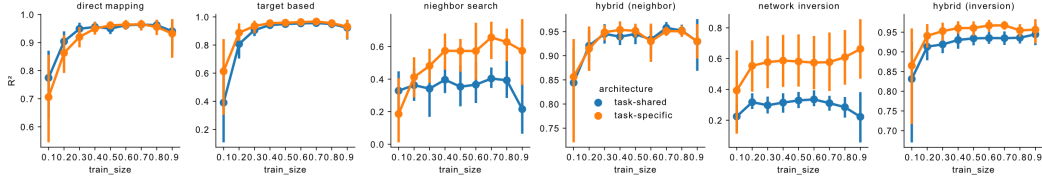

Figure 3: Effect of foundation model architecture on the performance for different training strategies. (task-shared: model with shared backbone and a common prediction head for all tasks, task-specific: model with shared backbone and task-specific heads)

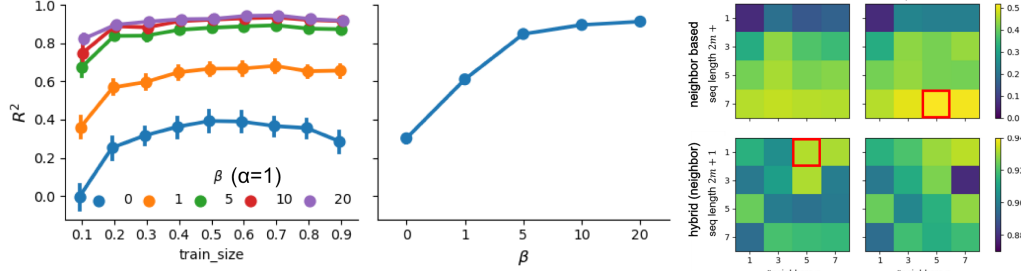

Figure 4: (Left) Effect of weightage $\beta$ of the target-based loss on the performance of models with task-specific prediction heads trained with hybrid strategy for refurbishing. The weightage $\alpha$ of input-based loss is set to be 1. The blue curve on the left ($\alpha = 1$, $\beta = 0$) represents purely neighbor-based refurbishing. (Right) Effect of sequence length $2m + 1$ and number of nearest neighbors $n$ on the performance of models with shared and task-specific prediction heads trained using input-based and hybrid strategies. For hybrid strategy, $\alpha = 1$ and $\beta = 20$ was selected. The combination which gave the best prediction performance in each case is marked with a red square (Please note that there exists multiple combinations which gave similar accuracies).

from impaired inputs without any training. Since no training was involved, it was not possible to report scores for cross-mapping with different training ratios.

**Direct-mapping.** In direct mapping, we used a refurbish module $h(.)$ in front of the pretrained model $g(.)$. The model learns to directly map the impaired individual's inputs $X_{amp}$ to the desired motion variables $\tilde{y}_{amp}$. During learning, the pretrained model is frozen, whereas the refurbish module is tunable.

**Fine-tuning (transfer learning).** In finetuning, no refurbish module is used, and the foundation module $g(.)$ pre trained on able-bodied data is finetuned to learn a mapping from impaired individual inputs $X_{amp}$ to the desired motion variables $\tilde{y}_{amp}$.

## 5.2 Ablation results

**Foundation module architectures.** Our initial analysis focused on how the architecture of the foundation module impacts prediction performance across various training strategies. As depicted in Fig. 3, we examined the foundation modules equipped with task-specific and task-shared prediction heads in predicting desired motion variables for mobility-impaired individuals across different tasks. For direct mapping and refurbished inputs with target-based and hybrid correction, models featuring task-specific heads outperformed those with shared prediction heads when the training data was limited. However, as the training dataset size increased, the performance of both models converged to similar levels. Interestingly, a reverse trend emerged when mobility-impaired individual inputs were refurbished with a target-based trained refurbish module. Models with shared prediction heads performed better for the smallest training data sizes, while those with task-specific layers performed better with larger training datasets. In summary, task-specific foundation models consistently demonstrated superior prediction performance when compared to their task-shared counterparts.

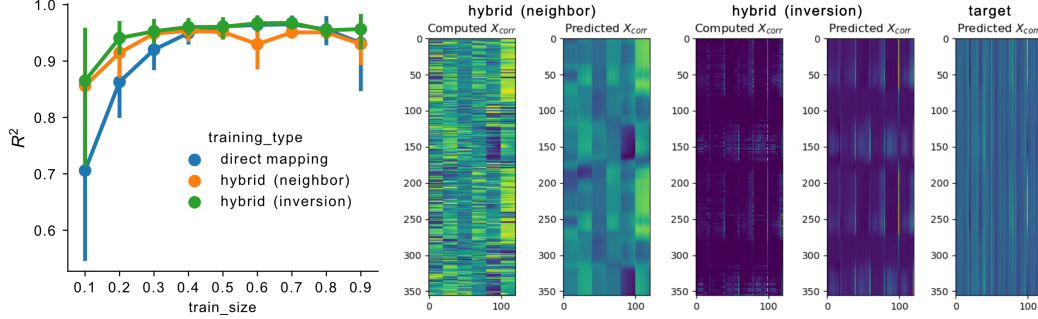

Figure 5: (Left) Performance of models trained with different strategies. For hybrid strategies, $\alpha = 1$ and $\beta = 20$ was selected. For neighbor-based strategies, $m$ and $n$ were selected based on the best-performing values computed in the previous section. (Right) The correction template $X_{corr}$ computed using different strategies and the corresponding predictions from the refurbish module. For target-based strategy, no correction template was computed, and the visualization shows the output of the refurbish module in this case.

**Effect of weightage $\beta$ of target-based loss .** In the hybrid training strategy of the refurbish module, we adopted a loss function that is a weighted sum of the correction-based and target-based loss. We next analyzed how the weights assigned to the loss functions affect the performance of the task-specific foundation modules. We set the weightage for correction-based loss to be 1 and varied the weightage $\beta$ of the target-based loss. We found that the model performance increases for higher weightage of target-based loss for refurbish model training. However, as the weightage $\beta$ increases, the prediction performance shows a converging trend (Fig. 4 left).

**Effect of sequence length $2m + 1$ and number of neighbors $n$.** In the nearest neighbor search method for computing the correction template $X_{corr}$, we proposed a strategy for using a sequence of desired motion variables for the mobility-impaired individual $\{y_{amp}^{k-m}, ..., y_{amp}^{k}, ..., y_{amp}^{k+m}\}$ to compute the correction template $X_{corr}^{k}$ for the $k$-th mobility-impaired individual input $X_{amp}^{k}$. Another strategy that we propose is to use a $\epsilon$-neighborhood around the able-bodied input sample $X_{ab}^{i}$ that is closest to the input required to produce the desired mobility-impaired individual motion variable $y_{amp}^{k}$ at time point $k$. The correction template should be computed as a weighted sum of $n$ nearest neighbors of $X_{ab}^{i}$ within the $\epsilon$-neighborhood (where weights decrease linearly or exponentially as the distance from $X_{ab}^{i}$ increases). We investigated the effect of the sequence length $2m + 1$ and number of nearest neighbors $n$ used for computing the correction template on the performance of models in predicting the desired motion variables for mobility-impaired individuals across locomotion tasks (Fig. 4 right). Most of the cases evaluated worked better than the basic case of single sample matching and single neighbor ($2m + 1 = 1, n = 1$). For models with task-shared prediction heads, better performances were obtained with lower sequence length $2m + 1$ and higher number of neighbors $n$ for both correction-based and hybrid methods. For models with task-specific prediction heads, correction-based correction performed better with larger sequence length and larger number of neighbors lead to better performance for whereas hybrid strategy worked better with a smaller sequence length and larger number of neighbors. Interestingly, hybrid (neighbor) strategy did not require matching a larger sequence of outputs, possibly because it also makes use of target-based correction. Nevertheless, sampling multiple neighbors from the $\epsilon$-neighborhood of $X_{ab}^{i}$ was still necessary to improve the prediction performance. A exponentially decreasing weighting performed better for neighbor-based correction strategy, whereas a linearly decreasing weighting worked better for hybrid strategy. For further analyses and comparisons, we use the sequence length, number of neighbors' and weighting values that gave the best prediction scores with neighbor-based and hybrid (neighbor) computed correction templates.

**Training strategies.** Finally, we examined the impact of various training strategies on model performance (Fig. 5 left and Tab. 1). Across all these strategies, we observed a general improvement in prediction scores as the training sample size increased. There was a slight drop in accuracy when using very large training sample sizes, likely due to overfitting. Both the hybrid strategies – hybrid (neighbor) and hybrid (inversion) — which combined correction-based and target-based mapping, consistently delivered accurate predictions across different training sample sizes. Notably, for the

Table 1: Coefficient of determination ($R^2$) obtained with different training strategies for a train sample ratio of 0.1

|  | cross mapping | direct mapping | fine tuning | neighbor search | network inversion | target based | hybrid (neighbor) | hybrid (inversion) |
|---|---|---|---|---|---|---|---|---|
| task-shared | -0.35 ± 0.16 | 0.77 ± 0.2 | 0.73 ± 0.22 | 0.33 ± 0.11 | 0.23 ± 0.07 | 0.39 ± 0.25 | **0.84 ± 0.07** | **0.83 ± 0.09** |
| task-specific | -0.32 ± 0.48 | 0.71 ± 0.14 | 0.74 ± 0.22 | 0.19 ± 0.22 | 0.39 ± 0.25 | 0.61 ± 0.22 | **0.86 ± 0.07** | **0.87 ± 0.09** |

smallest amount of training data tested, the hybrid strategy outperformed the direct mapping and fine-tuning approaches. With task-shared prediction heads, hybrid reprogramming showed a slight improvement ($\sim 7\%$) over direct mapping, while for task-specific prediction heads, it exhibited a more pronounced improvement ($\sim 16\%$). These results suggest that our proposed input refurbishing strategy offers an efficient means to adapt a pretrained model for new scenarios, especially when dealing with limited training data.

While direct mapping performs well in predicting motion variables for the specific mobility-impaired individual it was trained on, it requires a larger dataset to achieve the same performance level as the model reprogramming approach. Additionally, direct mapping relies on limited data from mobility-impaired individuals, reducing exposure to diverse motion conditions and leading to limited generalization capabilities. This limitation is critical for mobility-impaired individual motion prediction models, as gait patterns may change as mobility-impaired individuals adapt to generated motion. This adaptation could result in improved compensatory motions and gait normalization. In contrast, an able-bodied model trained on a wide array of motion scenarios from various individuals may be more adaptable to evolving mobility-impaired individual gait patterns while accommodating the predicted joint motion required for prosthetic walking. Our approach also outperformed model fine-tuning in the low-data regime, showing an improvement of approximately $\sim 11\%$ for task-shared models and $\sim 13\%$ for task-specific models, underlining the effectiveness of the proposed approach over other model repurposing methods like transfer learning.

Visualization of the computed correction templates $X_{corr}$ and corresponding predictions from the refurbish module (Fig. 5 right) shows that, despite the refurbish module being a simple three-layer MLP (chosen for its lightweight and data-efficient properties), it can reconstruct the correction templates with considerable accuracy. Interestingly, the correction templates computed by the neighbor search and network inversion strategies were notably different, despite having the same desired output. This discrepancy arises because neighbor search finds the closest match from available data points, introducing variability based on the dataset, while network inversion uses gradient-based optimization to iteratively minimize error, leading to different solutions. The same variability was observed with target-based mapping, which directly optimizes towards the target output without generating intermediate correction templates.

**Limitations** The desired motion variables for mobility-impaired individual subjects were derived from able-bodied individuals with similar anthropometric features and walking speed as that of the mobility-impaired individual subjects. While this approximation may effectively represent the reference joint's desired motion, its suitability for practical applications such as controlling powered prostheses requires validation. Nevertheless, our approach possesses a generic capability to transform inputs in a way that can generate any desired output from a pretrained model, provided there exists a learnable relationship between the inputs and the desired outputs.

# 6   Conclusion and Broader Impact

In this study, we proposed ReMAP, a model repurposing strategy that leverages deep learning's reprogramming property, incorporating network inversion principles and retrieval-augmented mapping. Our approach adapts models originally designed for able-bodied individuals to forecast joint motion in limb-impaired patients without altering model parameters. Our findings indicate that the proposed input refurbishing strategy offers a sample-efficient mechanism for adapting pretrained models to new scenarios. The proposed model reprogramming approach for adaptive motion forecasting has the potential to significantly enhance the quality of life for individuals with mobility impairments. By leveraging well-trained models from able-bodied data, this method can efficiently predict joint motions for mobility-impaired individuals, aiding in the development of more responsive and accurate

assistive devices such as prostheses and orthoses. This approach minimizes the need for extensive retraining, making it both cost-effective and accessible for personalized healthcare applications. Furthermore, this technique is versatile and can be applied to other regression problems in domains where data scarcity presents a challenge.

## Acknowledgements

The authors express their gratitude to Prof. Dr. Arndt F. Schilling (University Medical Center Göttingen, Georg-August University of Göttingen), Dr. Jennifer Ernst (University Medical Center Göttingen, Georg-August University of Göttingen), Dr. Massimo Sartori (University of Twente), Dr. Thomas Schmalz, Dr. Michael Ernst (Department of Clinical Research & Services, Ottobock SE & Co. KGaA, Germany), Dr. Takashi Yoshida (University Medical Center Göttingen, Georg-August University of Göttingen), and Dr.Mahdy Eslamy (Teesside University), for guidance, helping with experimental data collection involving lower-limb impaired subjects, and the writing of the ethics. The lower-limb impaired individuals' data collection was carried out under grant number INOPRO-16SV7657 supported by the Federal Ministry of Education and Research (BMBF) `https://www.interaktive-technologien.de/projekte/inopro`.

## Footnotes

\* Sharmita Dey conceptualized and led the project; both authors shared equal responsibility for executing the research.

## References

[1] Joerg Kindermann and Alexander Linden. Inversion of neural networks by gradient descent. *Parallel computing*, 14(3):277–286, 1990.

[2] Navid Ansari, Hans-Peter Seidel, Nima Vahidi Ferdowsi, and Vahid Babaei. Autoinverse: Uncertainty aware inversion of neural networks. *Advances in Neural Information Processing Systems*, 35:8675–8686, 2022.

[3] Pirzada Suhail, Supratik Chakraborty, and Amit Sethi. Network inversion of binarised neural nets. *arXiv preprint arXiv:2402.11995*, 2024.

[4] Wayne Xin Zhao, Jing Liu, Ruiyang Ren, and Ji-Rong Wen. Dense text retrieval based on pretrained language models: A survey. *ACM Transactions on Information Systems*, 42(4):1–60, 2024.

[5] Jiawei Zhou, Xiaoguang Li, Lifeng Shang, Xin Jiang, Qun Liu, and Lei Chen. Retrieval-based disentangled representation learning with natural language supervision. In *The Twelfth International Conference on Learning Representations*, 2023.

[6] George-Sebastian Pîrtoacă, Traian Rebedea, and Ștefan Rușeți. Improving retrieval-based question answering with deep inference models. In *2019 International Joint Conference on Neural Networks (IJCNN)*, pages 1–8. IEEE, 2019.

[7] Kailash A Hambarde and Hugo Proenca. Information retrieval: recent advances and beyond. *IEEE Access*, 2023.

[8] Gamaleldin F Elsayed, Ian Goodfellow, and Jascha Sohl-Dickstein. Adversarial reprogramming of neural networks. *arXiv preprint arXiv:1806.11146*, 2018.

[9] Yun-Yun Tsai, Pin-Yu Chen, and Tsung-Yi Ho. Transfer learning without knowing: Reprogramming black-box machine learning models with scarce data and limited resources. In *International Conference on Machine Learning*, pages 9614–9624. PMLR, 2020.

[10] Karen Hambardzumyan, Hrant Khachatrian, and Jonathan May. Warp: Word-level adversarial reprogramming. *arXiv preprint arXiv:2101.00121*, 2021.

[11] Chao-Han Huck Yang, Yun-Yun Tsai, and Pin-Yu Chen. Voice2series: Reprogramming acoustic models for time series classification. *arXiv e-prints*, pages arXiv–2106, 2021.

[12] Hao Yen, Pin-Jui Ku, Chao-Han Huck Yang, Hu Hu, Sabato Marco Siniscalchi, Pin-Yu Chen, and Yu Tsao. Neural model reprogramming with similarity based mapping for low-resource spoken command classification. In *Annual Conference of the International Speech Communication Association*, 2023.

[13] Tuan Dinh, Daewon Seo, Zhixu Du, Liang Shang, and Kangwook Lee. Improved input reprogramming for gan conditioning. *arXiv preprint arXiv:2201.02692*, 2022.

[14] Qizhou Wang, Feng Liu, Yonggang Zhang, Jing Zhang, Chen Gong, Tongliang Liu, and Bo Han. Watermarking for out-of-distribution detection. *Advances in Neural Information Processing Systems*, 35:15545–15557, 2022.

[15] Igor Melnyk, Vijil Chenthamarakshan, Pin-Yu Chen, Payel Das, Amit Dhurandhar, Inkit Padhi, and Devleena Das. Reprogramming pretrained language models for antibody sequence infilling. In *International Conference on Machine Learning*, pages 24398–24419. PMLR, 2023.

[16] Lingwei Chen, Yujie Fan, and Yanfang Ye. Adversarial reprogramming of pretrained neural networks for fraud detection. In *Proceedings of the 30th ACM International Conference on Information & Knowledge Management*, pages 2935–2939, 2021.

[17] Karen Simonyan, Andrea Vedaldi, and Andrew Zisserman. Deep inside convolutional networks: Visualising image classification models and saliency maps. *arXiv preprint arXiv:1312.6034*, 2013.

[18] Aravindh Mahendran and Andrea Vedaldi. Understanding deep image representations by inverting them. In *Proceedings of the IEEE conference on computer vision and pattern recognition*, pages 5188–5196, 2015.

[19] Alexey Dosovitskiy and Thomas Brox. Inverting visual representations with convolutional networks. In *Proceedings of the IEEE conference on computer vision and pattern recognition*, pages 4829–4837, 2016.

[20] Anh Nguyen, Jason Yosinski, and Jeff Clune. Multifaceted feature visualization: Uncovering the different types of features learned by each neuron in deep neural networks. *arXiv preprint arXiv:1602.03616*, 2016.

[21] Jun-Yan Zhu, Taesung Park, Phillip Isola, and Alexei A Efros. Unpaired image-to-image translation using cycle-consistent adversarial networks. In *Proceedings of the IEEE international conference on computer vision*, pages 2223–2232, 2017.

[22] Vladimir Karpukhin, Barlas Oğuz, Sewon Min, Patrick Lewis, Ledell Wu, Sergey Edunov, Danqi Chen, and Wen-tau Yih. Dense passage retrieval for open-domain question answering. *arXiv preprint arXiv:2004.04906*, 2020.

[23] Patrick Lewis, Ethan Perez, Aleksandra Piktus, Fabio Petroni, Vladimir Karpukhin, Naman Goyal, Heinrich Küttler, Mike Lewis, Wen-tau Yih, Tim Rocktäschel, et al. Retrieval-augmented generation for knowledge-intensive nlp tasks. *Advances in Neural Information Processing Systems*, 33:9459–9474, 2020.

[24] Gautier Izacard and Edouard Grave. Leveraging passage retrieval with generative models for open domain question answering. *arXiv preprint arXiv:2007.01282*, 2020.

[25] Erwin Aertbeliën and Joris De Schutter. Learning a predictive model of human gait for the control of a lower-limb exoskeleton. In *5th IEEE RAS/EMBS International Conference on Biomedical Robotics and Biomechatronics*, pages 520–525. IEEE, 2014.

[26] Fabian Horst, Sebastian Lapuschkin, Wojciech Samek, Klaus-Robert Müller, and Wolfgang I Schöllhorn. Explaining the unique nature of individual gait patterns with deep learning. *Scientific reports*, 9(1):2391, 2019.

[27] Sharmita Dey, Sabri Boughorbel, and Arndt F Schilling. Learning a shared model for motorized prosthetic joints to predict ankle-joint motion. *arXiv preprint arXiv:2111.07419*, 2021.

[28] Erika V Zabre-Gonzalez, Lara Riem, Philip A Voglewede, Barbara Silver-Thorn, Sara R Koehler-McNicholas, and Scott A Beardsley. Continuous myoelectric prediction of future ankle angle and moment across ambulation conditions and their transitions. *Frontiers in Neuroscience*, 15:709422, 2021.

[29] Zachary Choffin, Nathan Jeong, Michael Callihan, Edward Sazonov, and Seongcheol Jeong. Lower body joint angle prediction using machine learning and applied biomechanical inverse dynamics. *Sensors*, 23(1):228, 2022.

[30] Sharmita Dey and Arndt F Schilling. A function approximator model for robust online foot angle trajectory prediction using a single imu sensor: Implication for controlling active prosthetic feet. *IEEE Transactions on Industrial Informatics*, 19(2):1467–1475, 2022.

[31] Sharmita Dey, Mahdy Eslamy, Takashi Yoshida, Michael Ernst, Thomas Schmalz, and ArndtF Schilling. A support vector regression approach for continuous prediction of ankle angle and moment during walking: An implication for developing a control strategy for active ankle prostheses. In *2019 IEEE 16th International Conference on Rehabilitation Robotics (ICORR)*, pages 727–733. IEEE, 2019.

[32] Feng-Yan Liang, Fei Gao, Junyi Cao, Sheung-Wai Law, and Wei-Hsin Liao. Gait synergy analysis and modeling on amputees and stroke patients for lower limb assistive devices. *Sensors*, 22(13):4814, 2022.

[33] Minjae Kim and Levi J Hargrove. A gait phase prediction model trained on benchmark datasets for evaluating a controller for prosthetic legs. *Frontiers in Neurorobotics*, 16:1064313, 2023.

[34] Sharmita Dey, Takashi Yoshida, and Arndt F Schilling. Feasibility of training a random forest model with incomplete user-specific data for devising a control strategy for active biomimetic ankle. *Frontiers in Bioengineering and Biotechnology*, 8:855, 2020.

[35] Blair Hu, Elliott Rouse, and Levi Hargrove. Benchmark datasets for bilateral lower-limb neuromechanical signals from wearable sensors during unassisted locomotion in able-bodied individuals. *Frontiers in Robotics and AI*, 5:14, 2018.

[36] Sharmita Dey, Takashi Yoshida, Robert H Foerster, Michael Ernst, Thomas Schmalz, Rodrigo M Carnier, and Arndt F Schilling. A hybrid approach for dynamically training a torque prediction model for devising a human-machine interface control strategy. *arXiv preprint arXiv:2110.03085*, 2021.

[37] Scott L Delp, Frank C Anderson, Allison S Arnold, Peter Loan, Ayman Habib, Chand T John, Eran Guendelman, and Darryl G Thelen. Opensim: open-source software to create and analyze dynamic simulations of movement. *IEEE transactions on biomedical engineering*, 54(11):1940–1950, 2007.

[38] Shaojie Bai, J Zico Kolter, and Vladlen Koltun. An empirical evaluation of generic convolutional and recurrent networks for sequence modeling. *arXiv preprint arXiv:1803.01271*, 2018.

# A Appendix

## A.1 Network inversion

Starting with a random input $X^{(0)} \sim \mathcal{U}(0,1)$, the gradient of loss, $\mathcal{L}(g(X), \tilde{y}_{amp})$, of the foundation module output is computed with respect to its input $X$. At each iteration $t$, the input is updated using gradient descent,

$$X^{(t+1)} = X^{(t)} + \eta \frac{\partial \mathcal{L}}{\partial X^{(t)}} \tag{11}$$

where $\eta$ is the learning rate and $\mathcal{L}$ is mean-squared error loss function, until convergence condition, $\mathcal{L}(g(X), \tilde{y}_{amp}) < \tau$, is satisfied. We used $\eta = 0.01$, $\tau = 1e - 4$, and $\lambda = 1$.

## A.2 Theoretical basis

In line with [11], we define the population risk for the target task (mobility-impaired individual's motion prediction) via reprogramming a pretrained source network (able-bodied model) to be upper bounded by the sum of two terms:

1. **Source population risk**: The risk associated with the source task, which is denoted as $R_s(f)$.

2. **Representation alignment loss**: The Wasserstein-1 distance between the distributions of the source data (computed able-bodied template, $X^s$) and the reprogrammed target data ($h(X^t)$).

**Theorem Statement:** Let $h$ denote the learned additive input transformation for reprogramming and $f$ is the pretrained model. The population risk for the target task $R_t(h)$ is upper bounded by:

$$R_t(h) \leq R_s(f) + W_1(X^s, h(X^t)) \tag{12}$$

However, the second term in the risk function is a strict constraint and a slight deviation in the representational alignment can lead to a large error in the output. The network-inversion based loss function relaxes the constraint such that the reprogrammed target data no longer needs to be similar to the source data. However, the constraint now changes to a representational similarity between reprogrammed target data and network inversion inputs such that

$$R_t(h) \leq R_s(f) + W_1(f^{-1}(y^s), h(X^t)). \tag{13}$$

On the other hand, adding the target-based loss effectively relaxes this constraint by depending more on the source population risk and less on the stricter constraint of representational similarity of reprogrammed target data and source data. The target population risk becomes

$$R_t(h) \leq \beta R_s(f) + \alpha W_1(f^{-1}(y^s), h(X^t)). \tag{14}$$

with $\alpha + \beta = 1$ and $\alpha < \beta$. Since the source population risk depends on the performance of the pretrained model $f$, the population risk of the hybrid approach can effectively approach a lower error bound than 12 and 13.

## A.3 Experimental details

**Foundation module.** The foundation module, $g$, consists of a shared core $g_s$ and lightweight task-specific prediction heads $g_t$. The shared core consists of two time convolution layers, with a kernel size of 5 and dropout rate of 0.2. Each prediction head is a lightweight two-layer MLP with 200 and 100 units, with ReLU activation in hidden layers. The foundation module was trained using able-bodied data from a publicly available dataset [35] with a batch size of 100 for 25 epochs with stochastic gradient descent optimizer (learning rate of 1e-3 and momentum=0.9).

**Refurbish module.** Since we require the refurbish module to be data-efficient, a simple three-layer MLP with 100 units each with ReLU activation was used. The model is trained for 100 epochs with a

batch size of 100. A stochastic gradient descent optimizer with a learning rate of 1e-4 and momentum of 0.9 was used.

**Compute resources.** All experiments were compatible on single GPU (Nvdia GTX3070, 8GB) with 12 CPU cores (Intel i7).

## A.4 Extended results

**Optimal Beta value** We evaluated the performance of models for different beta values and found that the best performance was obtained with $\beta = 20$ which was eventually selected. The performance saturates at a $\beta$ of 20 and diminishes for larger values of $\beta$.

| $\beta$ | 1 | 5 | 10 | 20 | 30 | 40 | 50 |
|---------|------|------|------|------|------|------|------|
| $R^2$ | 0.63±0.04 | 0.88±0.02 | 0.92±0.02 | 0.94±0.01 | 0.94±0.02 | 0.90±0.05 | 0.85±0.07 |

**Correction-based mapping** In the below table, we report the correction-based mapping results for different training ratios. It can be observed that for small amount of training data, the correction-based method does not work well. However, as the amount of training data, the correction-based performance also increases (as the refurbish module becomes increasingly accurate). These results show that the performance of the correction-based mapping depends strongly on the accuracy of the refurbish module. By adding the target-based loss, this constraint is relaxed and the model now depends more on the performance of the pretrained model.

| Train size | 0.1 | 0.2 | 0.3 | 0.4 | 0.5 | 0.6 | 0.7 | 0.8 | 0.9 |
|------------|-----|-----|-----|-----|-----|-----|-----|-----|-----|
| Output $R^2$ | 0.18±0.14 | 0.41±0.12 | 0.48±0.11 | 0.58±0.11 | 0.57±0.11 | 0.57±0.16 | 0.66±0.09 | 0.63±0.07 | 0.57±0.17 |
| Refurbish model $R^2$ | 0.20±0.06 | 0.38±0.05 | 0.42±0.06 | 0.46±0.05 | 0.45±0.05 | 0.48±0.06 | 0.48±0.09 | 0.48±0.06 | 0.45±0.09 |

## A.5 User study

All the user studies presented here were approved by the Institutional Review Board (IRB) of the University Medical Center Göttingen. The participants gave their written consent to the study. Below you can find the snapshots of the general instructions given in the participant study, potential risks, precautions, and details about compensation.

# General instructions' snippets of relevant information given to participants during locomotion experiments (both English-translated and German version below)

Dear Study Participant,

     We hereby ask you to participate in our research study, carried out in the Applied Surgical and Rehabilitation Technology Lab, Department of Trauma Surgery, Orthopaedics and Plastic Surgery. Your participation is voluntary and you can take back your consent at any time without giving any reason and without any influence on your current or future medical treatments.

     Before submitting your consent, you have the right to be fully informed about the study.

Sehr geehrte Studienteilnehmerin, sehr geehrter Studienteilnehmer,

     hiermit bitten wir Sie um die Teilnahme an einem unserer Forschungsprojekte, die im Applied Surgical and Rehabilitation Technologies Lab der Klinik für Unfallchirurgie, Orthopädie und Plastischen Chirurgie durchgeführt werden. Ihre Teilnahme ist freiwillig und Sie können Ihre Zustimmung jederzeit ohne Angabe von Gründen und ohne Einfluss auf derzeitige oder zukünftige Behandlungen widerrufen.

     Zur Abgabe Ihrer Einverständniserklärung haben Sie das Recht Ihre Entscheidung zu bedenken, sowie eine Vertrauensperson während der mündlichen Aufklärung hinzuzuziehen.

## Goal of the study

After a leg amputation, the amputated leg can be replaced by a prosthesis. But, we do not know what leads to the best use of a prosthesis. The use of a prosthesis can be influenced by surgical techniques, functions of the prosthesis, and the type of rehabilitation. In this study, we aim to find out what happens to the way people move after a leg amputation. For this purpose, we will examine how people with an amputated leg perform tasks, which are related to walking, standing, and sitting. For comparison, we will examine people that have not experienced amputation during the same tasks. In this study, we also aim to find out what happens to a leg after an amputation. For this purpose, we will examine the amputated leg in detail. Lastly, to better understand your situation, we will ask you to complete questionnaires about your amputation (if applicable), quality of life, and how physically active you are. All of the methods in this study are non-invasive.

segments, and range of motion. We will also attach electrodes and markers to your body for Parts 3 and 4. In Parts 3 and 4 (Movement Examination 1 and 2), we will ask you to perform tasks that involve sitting, standing, and walking. You have the right to refuse any tasks.

Part 5 (Physical Examination 2) will be performed on a different day from Parts 2, 3, and 4. Part 5 will only be performed for people with amputated legs. In Part 5, we want to examine whether people can feel gentle stimulation of the amputated and non-amputated legs. The gentle stimulation will include light touch, gentle vibrations, light pressure, and low-intensity electrical stimulation. All forms of stimulation are non-invasive.

| Part 1: Interview | Part 2: Physical Examination 1 | Part 3: Movement Examination 1 | Part 4: Movement Examination 2 | Part 5: Physical Examination 2 |
|---|---|---|---|---|
| **Questions** about your health, medications, amputation, prosthesis<br><br>**Questionnaires** about your quality of life and mobility | **Height**<br><br>**Weight**<br><br>**Length of body segments**<br><br>**Range of motion** at the hip, knee, and ankle<br><br>**Photographs**<br><br>Attach **markers** and **electrodes** for Parts 3 and 4 | **Standing**<br><br>**Maximum contraction** of muscles<br><br>**Sitting** down and **standing** up | **Walking** on a **treadmill**<br><br>**Walking** on a **flat surface**<br>a) Comfortable walking speed<br>b) Faster walking speed<br>c) Slower walking speed<br><br>**Walking** on **ramps**<br><br>**Walking** on **stairs** | **Palpation** of the amputated leg<br><br>Examination of **sensations** and **perfusion** of the amputated leg<br><br>**Measurements** of the amputated leg<br><br>**Ultrasound** of the amputated leg<br><br>**Photographs** |
| **45 Minutes** | **90 Minutes** | **30 Minutes** | **155 Minutes** | **145 Minutes** |
| **Regular breaks are planned and breaks on demand can be inserted at any time.** | | | | |

## Ziel der Studie

Eine Prothese ersetzt nach einer Amputation verlorene Funktionen des verlorenen Teils des Beins, wie zum Beispiel die Fähigkeit zu Gehen. Welche weiteren Auswirkungen durch die Amputation auf benachbarte Gelenke und den Amputationsstumpf auftreten oder welche Prothesenfunktionen oder Rehabilitationsstrategien zu einer optimalen Benutzung der Prothese führen ist unklar.

Daher ist das Ziel dieser Studie die Beurteilung typischer Veränderungen, die nach einer Amputation im Gangbild und am Stumpf auftreten. Hierfür werden Menschen, die amputiert sind im Ganglabor beim Gehen untersucht. Zudem wollen wir Ihren Amputationsstumpf und das gesunde Bein untersuchen und vergleichen. Untersucht werden zum Beispiel Empfindungen am Stumpf (z.B. kalt, warm, Schmerzen, Missempfindungen) oder Versorgung der Haut des Stumpfes mit Sauerstoff. Zum besseren Verständnis Ihrer Situation werden wir Ihnen Fragen stellen und Fragebögen austeilen.

Die Studie ist in 5 Teile aufgeteilt. Die Dauer der Einzelteile der Studie können Sie aus der Tabelle entnehmen. Am Anfang jedes Untersuchungsteils gibt es eine kurze Einführung in das Protokoll und mögliche Risiken der Prozedur. Zu diesem Zeitpunkt sowie während des ganzen Versuchs haben die Probanden die Möglichkeit, Fragen zu stellen und das Experiment abzubrechen.

Bei den beschriebenen Methoden handelt es sich um medizinische Standardverfahren, wie sie im täglichen Klinikbetrieb verwendet werden. Im ersten Teil (Interview) werden Sie befragt und gebeten Fragebögen zu vervollständigen, die wir Ihnen vorab per Post zugestellt haben. Die Teile 2, 3 und 4 werden am selben Tag durchgeführt. Dann werden einige Untersuchungen (körperliche Untersuchung 1) zu Ihrer Körperstatik an Ihnen im Liegen und Stehen durchgeführt.

Im dritten und vierten Teil des Versuchs (aktive Untersuchung 1&2) werden Winkelsensoren und reflektierende Marker an Ihnen befestigt, durch die wir mit

Kameras Ihr Bewegungen aufzeichnen und genau nachvollziehen können wie sie mit Ihrer Prothese gehen.

Teil 5 wird an einem anderen Tag als Teil 2, 3 und 4 durchgeführt. Teil 5 wird nur für Menschen mit amputierten Beinen durchgeführt. Im letzten, dem fünften Teil des Versuchs (körperliche Untersuchung 2) möchten wir untersuchen, ob an bestimmten Stellen der Hautoberfläche des Stumpfs Empfindungen für das amputierte Gliedmaß ausgelöst werden können. Dies wird durch leichte Berührung mit Wattestäbchen systematisch untersucht. In einem zweiten Schritt werden zwei Arten taktiler Stimulation benutzt, um den Tastsinn anzuregen. Einerseits werden durch kleine Elektromotoren sanfte Vibrationen und/oder ein leichter Druck auf der Haut erzeugt. Diese Elektromotoren werden mit einem elastischen Band am Stumpf und dem gesunden Bein angebracht. Dieser Vorgang ist vollkommen schmerzfrei und ohne bleibende Schäden. Als andere Stimulationsart wird eine elektrische Stimulation mit Hilfe kleiner Elektroden erzeugt. Die selbstklebenden Elektroden werden auf der Hautoberfläche angebracht. Diese Art der Stimulation wird als „elektrokutane Stimulation" bezeichnet, weil ein Strom geringer Intensität oberflächlich die Haut stimuliert. Die ausgelösten Empfindungen können einer Kombination des Klopfens, Drucks, und/oder des Vibrierens entsprechen. Bis zu 16 kleine Motoren werden an Gliedmaß- und Stumpfarealen angebracht. Die beiden erwähnten Methoden, d.h. elektrische und mechanische Stimulation, beruhen auf allgemein verwendeten Methoden, um gefühlsmäßige Rückkopplungssignale im geschlossenen Regelkreis zu

liefern. Diese Methoden wurden bereits in mehreren vergangenen Studien sicher angewendet. In dieser Studie möchten wir testen, wie empfindlich die Probanden an den stimulierten Stellen sind. Genauer geht es darum unterschiedlich starke Aktivierungen einzuordnen (kaum spürbar, klar wahrnehmbare und fast unangenehme Vibrationen).

Wir würden uns sehr freuen, wenn Sie die Fragen und Fragebögen während unseres Versuchs zu Ihrer Person und Ihrer körperlichen Verfassung beantworten

würden. Wenn Ihnen eine Frage unangenehm ist, steht es Ihnen selbstverständlich frei, darauf nicht zu antworten.

| (1) Interview | (2) körperliche Untersuchung 1 | (3) Aktive Untersuchung 1 | (4) Aktive Untersuchung 2 | (5) körperliche Untersuchung 2 |
|---|---|---|---|---|
| **Fragen** zu Ihrer Gesundheit, Medikamenten, die Sie nehmen, Amputation und Ihrer Prothese<br><br>**Fragebögen** zu Ihrer Lebensqualität und Mobilität | **Größe Gewicht**<br><br>**Körperstatik** und **Haltung Messen** der Körperteillängen<br><br>**Bewegungs-freiheit** der Hüft-, Knie- und Sprung-gelenke<br><br>**Foto**dokumentation<br><br>**Anbringen** der **Marke**r für Teil 3,4 | Ruhiges **Stehen**<br><br>Maximale **Muskelkraft** in Fuß und Bein<br><br>**Hinsetzen & Aufstehen** | **Gehen** auf einem **Laufband**<br><br>**Gehen** auf einer **ebenen Fläche** a) selbstgewählte Gehgeschwindigkeit b) Schnellere Gehgeschwindigkeit c) Langsamere Gehgeschwindigkeit<br><br>**Gehen** auf einer **Rampe**<br><br>**Treppen** gehen | **Abtasten** des Stumpfs<br><br>**Untersuchung** von Empfindungen und Durchblutung am Stumpf<br><br>**Vermessen** des Stumpfs<br><br>**Ultraschall** des Stumpfs, **Foto**dokumentation Stumpf **(2)** |
| 45 Minuten | 90 Minuten | 30 Minuten | 155 Minuten | 145 Minuten |
| **Regelmäßige Pausen sind geplant und Pausen bei Bedarf können jederzeit eingefügt werden** | | | | |

## Risiken

Die Risiken der zuvor beschriebenen Schritte sind minimal.

Die nicht-invasiven Schritte zur Applizierung der Oberflächenelektroden (Stimulation und EMG) können zu leichtem Unbehagen durch Abrieb an der Haut führen. Das zur Verbesserung des Haut-Elektroden-Kontaktes verwendete Hautreinigungsgel kann in sehr seltenen Fällen zu allergischen Reaktionen in Form von lokalen Rötungen führen. Manche Probanden können die Stimulation mit leichtem Strom/Vibration/Druck als unangenehm empfinden. Speziell für den Fall der Elektrostimulation kann die Haut unter der Stimulationselektrode eine leichte Rötung aufweisen. Nach Unterbrechung der Stimulation ist die Rötung im Normalfall nach 30-60 Minuten nicht mehr vorhanden. Selten kann eine allergische Reaktion durch die Elektrostimulation entstehen (lokale Schwellung oder Rötung). Wenn die Elektrostimulation nicht ordnungsgemäß durchgeführt wird, kann diese als

schmerzhaft empfunden werden. Unser Versuchsaufbau verwendet Standardverfahren mit minimaler Stimulationsintensität, wodurch dieses Risiko minimiert wird.

## Personal data

During the study, all of the data that we collect from you will be pseudonymized. This means that your data will be labeled with a randomly generated code, not with your name. Only the study investigators will know which code applies to your data. Also, your face will not appear in any of the photographs that we take.

For this study, we are partnering with Otto Bock GmbH (Germany) and the University of Twente (The Netherlands). Our partners will only receive anonymized data. This means that our partners cannot trace the data to your identity. For research

purposes, we may send the collected data to third parties (people or institutions that are not the study investigators). Third parties will only receive anonymized data. For publishing our findings in academic journals, we will only use anonymized data.

If you decide to take back your consent or withdraw from the study, your data will be deleted. After this study is terminated, the pseudonymized data will be stored for 10 years.

**Persönliche Daten**

Während der Studie werden personenbezogene Gesundheitsdaten erhoben und in pseudonymisierter Form aufgezeichnet. Pseudonymisierung bedeutet Verschlüsselung von Daten ohne Namensnennung nur mit Nummern codiert. Die Zuordnung der Daten oder Proben zu einer Person ist nur möglich, wenn hierfür der Schlüssel eingesetzt wird, mit dem die Daten pseudonymisiert wurden. Die personenbezogenen Daten/Proben werden unter besonderen Schutzvorkehrungen getrennt von der Schlüsselliste aufbewahrt. Eine Entschlüsselung ist nur durch die für die Studie verantwortlichen Personen möglich. Dritte erhalten keinen Einblick in die Originalunterlagen. Die Datenspeicherungszeit beträgt 10 Jahre. Eine Datenweitergabe pseudonymisierter, das heißt verschlüsselter, Daten ist mit den Kooperationspartnern der Firma Otto Bock und der Universität Twente in den Niederlanden geplant. Dies betrifft Daten aller Teilexperimente. Dabei ist sichergestellt, dass der Forschungspartner die Daten nicht auf einzelne Personen rückverfolgen kann. Bei Widerruf der Studienteilnahme werden die personenbezogenen Daten gelöscht. Eine Veröffentlichung findet nur mit anonymen Daten statt. Die experimentellen Ergebnisse können einer dritten Person für Forschungszwecke weiter gegeben werden. In diesem Fall werden die Daten verschlüsselt und der Privatbereich der Teilnehmer völlig geschützt sein.

**Participating in the Study**

To participate in this study, you must be willing and legally able to provide your written informed consent. Your eligibility will be checked by a medical doctor. If any of the following conditions applies to you, you cannot participate in the study:

- Medical conditions that cause visibly abnormal movement
  - Examples: Broken bones, pain
- A major amputation other than an above-knee, below-knee, or through-knee amputation on one side
- Dementia
- Open wounds, swelling, infection, or pain in the amputated leg
- Risk of stress fractures or fractures from low-impact falls
- Pregnancy
- Addiction to drugs or alcohol
- Inability to communicate with the investigators
- Lack of cooperation with the investigators

- Medical conditions that make participation dangerous
  - Example: cardiovascular disease

## Teilnahme an der Studie

Voraussetzung für die Teilnahme an der Studie ist die Einwilligung und Einwilligungsfähigkeit. Die Ein- und Ausschlusskriterien werden von einem Arzt

überprüft. Sollte eine der folgenden Punkte auf Sie zutreffen, ist eine Teilnahme an der Studie nicht möglich:

## Premature Interruption of the Study

If you experience any discomfort during the study, you have the right to stop participating in the study. If you develop any condition that makes you ineligible for the study while you are participating, the investigators will stop your participation. You have the right to stop your participation at any time for any reason. If your participation is stopped prematurely, you will be paid for the time you have spent.

## Approval of the Study

All procedures for this study have been approved by the responsible ethics committee under the processing number XXXXXX.

## Reimbursement

You will be reimbursed at €10 per hour for your participation. This includes the time required for transportation to and from the study.

## Further Questions

If you have any further questions, please do not hesitate to contact us.

Your Study Team

**Abbruch der Studie**

Falls bei Ihnen unerwartet Rötungen, allergische Reaktionen, Herz- Kreislauf Symptome während des Versuches auftreten oder Sie im Verlauf Ausschlusskriterien aufweisen, dann kann jeder einzelne Versuchsschritt jederzeit abgebrochen werden. Sie erhalten dann eine Bezahlung entsprechend dem Zeitaufwand, an dem Sie tatsächlich teilgenommen haben. Sie können zu jedem Zeitpunkt die Teilnahme am Forschungsprojekt abrechen. Natürlich würden wir dann gerne von Ihnen wissen wollen, warum Sie die Teilnahme abbrechen. Aber Sie brauchen uns keine Gründe nennen, wenn Sie nicht möchten.

**Genehmigung des Versuchs**

Die Versuchsdurchführung, die wir Ihnen hier vorstellen, wurde von der zuständingen Ethik-Kommission unter der Bearbeitungsnummer 26/3/18 zustimmend bewertet.

**Aufwandsentschädigung**

Zur Aufwandsentschädigung für die Teilnahme an der Studie erhalten Sie einen Betrag von 10 €/h. Im Falle eines vorzeitigen Abbruchs der Studie erhalten Sie ebenfalls eine Aufwandsentschädigung, entsprechend der bis dahin beanspruchten Zeit. Die Anreisezeit wird ebenfalls mit 10 €/h entschädigt.

**Weitere Fragen**

Sollten Sie weitere Fragen haben, zögern Sie nicht die Prüfdurchführenden anzusprechen.

Bei Fragen stehen wir Ihnen gerne zur Verfügung.

Ihr Studien-Team

